# Learning to Parse Images

**Geoffrey E. Hinton and Zoubin Ghahramani**
Gatsby Computational Neuroscience Unit
University College London
London, United Kingdom WC1N 3AR
*{hinton,zoubin} @gatsby.ucl.ac.uk*

**Yee Whye Teh**
Department of Computer Science
University of Toronto
Toronto, Ontario, Canada M5S 3G4
*ywteh@cs.utoronto.ca*

## Abstract

We describe a class of probabilistic models that we call credibility networks. Using parse trees as internal representations of images, credibility networks are able to perform segmentation and recognition simultaneously, removing the need for *ad hoc* segmentation heuristics. Promising results in the problem of segmenting handwritten digits were obtained.

## 1 Introduction

The task of recognition has been the main focus of attention of statistical pattern recognition for the past 40 years. The paradigm problem is to classify an object from a vector of features extracted from the image. With the advent of backpropagation [1], the choice of features and the choice of weights to put on these features became part of a single, overall optimization and impressive performance was obtained for restricted but important tasks such as handwritten character identification [2].

A significant weakness of many current recognition systems is their reliance on a separate preprocessing stage that segments one object out of a scene and approximately normalizes it. Systems in which segmentation precedes recognition suffer from the fact that the segmenter does not know the shape of the object it is segmenting so it cannot use shape information to help it. Also, by segmenting an image, we remove the object to be recognized from the context in which it arises. Although this helps in removing the clutter present in the rest of the image, it might also reduce the ability to recognize an object correctly because the context in which an object arises gives a great deal of information about the nature of the object. Finally, each object can be described in terms of its parts, which can also be viewed as objects in their own right. This raises the question of how fine-grained the segmentations should be. In the words of David Marr : "Is a nose an object? Is a head one? ... What about a man on a horseback?" [3].

The successes of structural linguistics inspired an alternative approach to pattern recognition in which the paradigm problem was to parse an image using a hierarchical grammar of scenes and objects. Within linguistics, the structural approach was seen as an advance over earlier statistical approaches and for many years linguists eschewed probabilities, even though it had been known since the 1970's that a version of the EM algorithm could be used to fit stochastic context free grammars. Structural pattern recognition inherited the linguists aversion to probabilities and as a result it never worked very well for real data. With the advent of graphical models it has become clear that structure and probabilities can coexist. Moreover, the "explaining away" phenomenon that is central to inference in directed acyclic graphical models is exactly what is needed for performing inferences about possible segmentations of an image.

In this paper we describe an image interpretation system which combines segmentation and recognition into the same inference process. The central idea is the use of parse trees of images. Graphical models called credibility networks which describe the joint distribution over the latent variables and over the possible parse trees are used. In section 2 we describe some current statistical models of image interpretation. In section 3 we develop credibility networks and in section 4 we derive useful learning and inference rules for binary credibility networks. In section 5 we demonstrate that binary credibility networks are useful in solving the problem of classifying and segmenting binary handwritten digits. Finally in section 6 we end with a discussion and directions for future research.

## 2   Related work

Neal [4] introduced generative models composed of multiple layers of stochastic logistic units connected in a directed acyclic graph. In general, as each unit has multiple parents, it is intractable to compute the posterior distribution over hidden variables when certain variables are observed. However, Neal showed that Gibbs sampling can be used effectively for inference [4]. Efficient methods of approximating the posterior distribution were introduced later [5, 6, 7] and these approaches were shown to yield good density models for binary images of handwritten digits [8]. The problem with these models which make them inappropriate for modeling images is that they fail to respect the 'single-parent' constraint : in the correct interpretation of an image of opaque objects each object-part belongs to at most one object – images need parse trees, not parse DAGs.

Multiscale models [9] are interesting generative models for images that use a fixed tree structure. Nodes high up in the tree control large blocks of the image while bottom level leaves correspond to individual pixels. Because a tree structure is used, it is easy to compute the exact posterior distribution over the latent (non-terminal) nodes given an image. As a result, the approach has worked much better than Markov random fields which generally involve an intractable partition function. A disadvantage is that there are serious block boundary artifacts, though overlapping trees can be used to smooth the transition from one block to another [10]. A more serious disadvantage is that the tree cannot possibly correspond to a parse tree because it is the same for every image.

Zemel, Mozer and Hinton [11] proposed a neural network model in which the activities of neurons are used to represent the instantiation parameters of objects or their parts, i.e. the viewpoint-dependent coordinate transformation between an object's intrinsic coordinate system and the image coordinate system. The weights on connections are then used to represent the viewpoint-invariant relationship between the instantiation parameters of a whole, rigid object and the instantiation parame-

ters of its parts. This model captures viewpoint invariance nicely and corresponds to the way viewpoint effects are handled in computer graphics, but there was no good inference procedure for hierarchical models and no systematic way of sharing modules that recognize parts of objects among multiple competing object models.

Simard *et al* [12] noted that small changes in object instantiation parameters result in approximately linear changes in (real-valued) pixel intensities. These can be captured successfully by linear models. To model larger changes, many locally linear models can be pieced together. Hinton, Dayan and Revow [13] proposed a mixture of factor analyzers for this. Tipping and Bishop have recently shown how to make this approach much more computationally efficient [14]. To make the approach really efficient, however, it is necessary to have multiple levels of factor analyzers and to allow an analyzer at one level to be shared by several competing analyzers at the next level up. Deciding which subset of the analyzers at one level should be controlled by one analyzer at the level above is equivalent to image segmentation or the construction of part of a parse tree and the literature on linear models contains no proposals on how to achieve this.

## 3   A new approach to image interpretation

We developed a class of graphical models called credibility networks in which the possible interpretations of an image are parse trees, with nodes representing object-parts and containing latent variables. Given a DAG, the possible parse trees of an image are constrained to be individual or collections of trees where each unit satisfies the single-parent constraint, with the leaves being the pixels of an image. Credibility networks describe a joint distribution over the latent variables and possible tree structures. The EM algorithm [15] can be used to fit credibility networks to data.

Let $i \in I$ be a node in the graph. There are three random variables associated with $i$. The first is a multinomial variate $\lambda_i = \{\lambda_{ij}\}_{j \in pa(i)}$ which describes the parent of $i$ from among the potential parents $pa(i)$ :

$$\lambda_{ij} = \begin{cases} 1 & \text{if parent of } i \text{ is } j, \\ 0 & \text{if parent of } i \text{ is not } j. \end{cases} \tag{1}$$

The second is a binary variate $s_i$ which determines whether the object $i$[1] is present ($s_i = 1$) or not ($s_i = 0$). The third is the latent variables $x_i$ that describe the pose and deformation of the object. Let $\Lambda = \{\lambda_i : i \in I\}, S = \{s_i : i \in I\}$ and $X = \{x_i : i \in I\}$.

Each connection $j \to i$ has three parameters also. The first, $c_{ij}$ is an unnormalized prior probability that $j$ is $i$'s parent given that object $j$ is present. The actual prior probability is

$$\pi_{ij} = \frac{c_{ij} s_j}{\sum_{k \in pa(i)} c_{ik} s_k} \tag{2}$$

We assume there is always a unit $1 \in pa(i)$ such that $s_1 = 1$. This acts as a default parent when no other potential parent is present and makes sure the denominator in (2) is never 0. The second parameter, $p_{ij}$, is the conditional probability that object $i$ is present given that $j$ is $i$'s parent ($\lambda_{ij} = 1$). The third parameter $t_{ij}$ characterizes the distribution of $x_i$ given $\lambda_{ij} = 1$ and $x_j$. Let $\theta = \{c_{ij}, p_{ij}, t_{ij} : i \in I, j \in pa(i)\}$.

Using Bayes' rule the joint distribution over $\Lambda, S$ and $X$ given $\theta$ is $p(\Lambda, S, X|\theta) = p(\Lambda, S|\theta)p(X|\Lambda, S, \theta)$. Note that $\Lambda$ and $S$ together define a parse tree for the image. Given the parse tree the distribution over latent variables $p(X|\Lambda, S, \theta)$ can be

efficiently inferred from the image. The actual form of $p(X|\Lambda, S, \theta)$ is unimportant. The joint distribution over $\Lambda$ and $S$ is

$$P(\Lambda, S|\theta) = \prod_{i \in I} \prod_{j \in pa(i)} \left(\pi_{ij} p_{ij}^{s_i} (1 - p_{ij})^{1 - s_i}\right)^{\lambda_{ij}} \tag{3}$$

## 4  Binary credibility networks

The simulation results in section 5 are based on a simplified version of credibility networks in which the latent variables $X$ are ignored. Notice that we can sum out $\Lambda$ from the joint distribution (3), so that

$$P(S|\theta) = \prod_{i \in I} \sum_{j \in pa(i)} \pi_{ij} p_{ij}^{s_i} (1 - p_{ij})^{1 - s_i} \tag{4}$$

Using Bayes' rule and dividing (3) by (4), we have

$$P(\Lambda|S, \theta) = \prod_{i \in I} \prod_{j \in pa(i)} \left( \frac{c_{ij} s_j p_{ij}^{s_i} (1 - p_{ij})^{1 - s_i}}{\sum_{k \in pa(i)} c_{ik} s_k p_{ik}^{s_i} (1 - p_{ik})^{1 - s_i}} \right)^{\lambda_{ij}} \tag{5}$$

Let $r_{ij} = c_{ij} p_{ij}^{s_i} (1 - p_{ij})^{1 - s_i}$. We can view $r_{ij}$ as the unnormalized posterior probability that $j$ is $i$'s parent given that object $j$ is present. The actual posterior is the fraction in (5) :

$$\omega_{ij} = \frac{r_{ij} s_j}{\sum_{k \in pa(i)} r_{ik} s_k} \tag{6}$$

Given some observations $\mathcal{O} \subset S$, let $\mathcal{H} = S \setminus \mathcal{O}$ be the hidden variables. We approximate the posterior distribution for $\mathcal{H}$ using a factored distribution

$$Q(\mathcal{H}) = \prod_{i \in I} \sigma_i^{s_i} (1 - \sigma_i)^{1 - s_i} \tag{7}$$

The variational free energy, $\mathcal{F}(Q, \theta) = E_Q[-\log P(S|\theta) + \log Q(S)]$ is

$$\mathcal{F}(Q, \theta) = \sum_{i \in I} \left( E_Q \left[ \log \sum_{j \in pa(i)} c_{ij} s_j - \log \sum_{j \in pa(i)} c_{ij} s_j p_{ij}^{s_i} (1 - p_{ij})^{1 - s_i} \right] \right) +$$
$$\sum_{i \in I} \left( \sigma_i \log \sigma_i + (1 - \sigma_i) \log (1 - \sigma_i) \right) \tag{8}$$

The negative of the free energy $-\mathcal{F}$ is a lower bound on the log likelihood of generating the observations $\mathcal{O}$. The variational EM algorithm improves this bound by iteratively improving $-\mathcal{F}$ with respect to $Q$ (E-step) and to $\theta$ (M-step). Let $ch(i)$ be the possible children of $i$. The inference rules can be derived from (8) :

$$\sigma_i = sigmoid \left( \begin{array}{c} E_Q \left[ \log \sum_{j \in pa(i)} c_{ij} s_j p_{ij} - \log \sum_{j \in pa(i)} c_{ij} s_j (1 - p_{ij}) \right] \\ + \sum_{l \in ch(i)} E_Q \left[ \log \sum_{j \in pa(l)} r_{lj} s_j - \log \sum_{j \in pa(l)} c_{lj} s_j \right]_{\sigma_i = 0}^{\sigma_i = 1} \end{array} \right) \tag{9}$$

Let $D$ be the training set and $Q_d$ be the mean field approximation to the posterior distribution over $\mathcal{H}$ given the training data (observation) $d \in D$. Then the learning

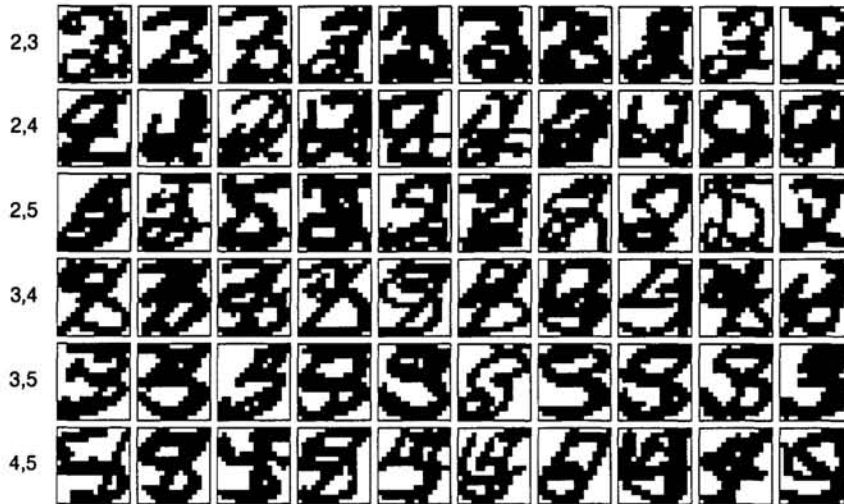

Figure 1: Sample images from the test set. The classes of the two digits in each image in a row are given to the left.

rules are

$$\frac{\partial - \sum_d \mathcal{F}(Q_d, \theta)}{\partial \log c_{ij}} = \sum_{d \in D} E_{Q_d}\left[\omega_{ij} - \pi_{ij}\right] \tag{10}$$

$$p_{ij}^{new} = \frac{\sum_{d \in D} E_{Q_d}[\omega_{ij} s_i]}{\sum_{d \in D} E_{Q_d}[\omega_{ij}]} \tag{11}$$

For an efficient implementation of credibility networks using mean field approximations, we still need to evaluate terms of the form $E[\log x]$ and $E[1/x]$ where $x$ is a weighted sum of binary random variates. In our implementation we used the simplest approximations : $E[\log x] \approx \log E[x]$ and $E[1/x] \approx 1/E[x]$. Although biased the implementation works well enough in general.

## 5   Segmenting handwritten digits

Hinton and Revow [16] used a mixture of factor analyzers model to segment and estimate the pose of digit strings. When the digits do not overlap, the model was able to identify the digits present and segment the image easily. The hard cases are those in which two or more digits overlap significantly. To assess the ability of credibility networks at segmenting handwritten digits, we used superpositions of digits at exactly the same location. This problem is much harder than segmenting digit strings in which digits partially overlap.

The data used is a set of 4400 images of single digits from the classes 2, 3, 4 and 5 derived from the CEDAR CDROM 1 database [17]. Each image has size 16x16. The size of the credibility network is 256-64-4. The 64 middle layer units are meant to encode low level features, while each of the 4 top level units are meant to encode a digit class. We used 700 images of single digits from each class to train the network. So it was not trained to segment images. During training we clamped at 1 the activation of the top layer unit corresponding to the class of the digit in the current image while fixing the rest at 0.

After training, the network was first tested on the 1600 images of single digits not in the training set. The predicted class of each image was taken to be the

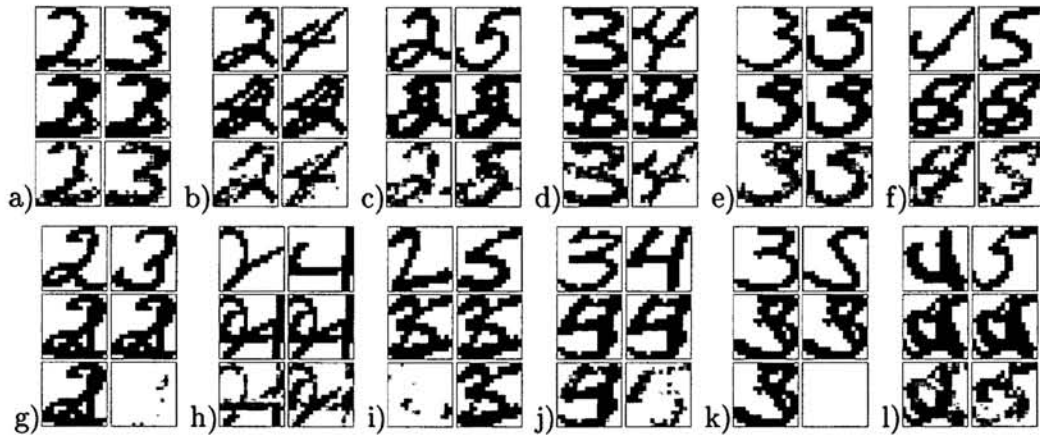

Figure 2: Segmentations of pairs of digits. (To make comparisons easier we show the overlapping image in both columns of a)-l).)

class corresponding to the top layer unit with the highest activation. The error rate was 5.5%. We then showed the network 120 images of two overlapping digits from distinct classes. There were 20 images per combination of two classes. Some examples are given in Figure 1. The predicted classes of the two digits are chosen to be the corresponding classes of the 2 top layer units with the highest activations. A human subject (namely the third author) was tested on the same test set. The network achieved an error rate of 21.7% while the author erred on 19.2% of the images.

We can in fact produce a segmentation of each image into an image for each class present. Recall that given the values of $S$ the posterior probability of unit $j$ being pixel $i$'s parent is $\omega_{ij}$. Then the posterior probability of pixel $i$ belonging to digit class $k$ is $\sum_j E_Q[\omega_{ij}\omega_{jk}]$.

This gives a simple way to segment the image. Figure 2 shows a number of segmentations. Note that for each pixel, the sum of the probabilities of the pixel belonging to each digit class is 1. To make the picture clearer, a white pixel means a probability of $\leq .1$ of belonging to a class, while black means $\geq .6$ probability, and the intensity of a gray pixel describes the size of the probability if it is between .1 and .6. Figures 2a) to 2f) shows successful segmentations, while Figure 2g) to 2l) shows unsuccessful segmentations.

## 6   Discussion

Using parse trees as the internal representations of images, credibility networks avoid the usual problems associated with a bottom-up approach to image interpretation. Segmentation can be carried out in a statistically sound manner, removing the need for hand crafted *ad hoc* segmentation heuristics. The granularity problem for segmentation is also resolved since credibility networks use parse trees as internal representations of images. The parse trees describe the segmentations of the image at every level of granularity, from individual pixels to the whole image.

We plan to develop and implement credibility networks in which each latent variable $x_i$ is a multivariate Gaussian, so that a node can represent the position, orientation and scale of a 2 or 3D object, and the conditional probability models on the links can represent the relationship between a moderately deformable object and its parts.

## Acknowledgments

We thank Chris Williams, Stuart Russell and Phil Dawid for helpful discussions and NSERC and ITRC for funding.

## Footnotes

[1]Technically this should be the object represented by node $i$.

## References

[1] D. E. Rumelhart, G. E. Hinton, and R. J. Williams. Learning internal representations by error propagation. In D. E. Rumelhart, J. L. McClelland, and the PDP Research Group, editors, *Parallel Distributed Processing : Explorations in The Microstructure of Cognition. Volume 1 : Foundations.* The MIT Press, 1986.

[2] Y. Le Cun, B. Boser, J. S. Denker, S. Solla, R. E. Howard, and L. D. Jackel. Back-propagation applied to handwritten zip code recognition. *Neural Computation*, 1(4):541–551, 1989.

[3] D. Marr. *Vision : A Computational Investigation into the Human Representation and Processing of Visual Information.* W. H. Freeman and company, San Francisco, 1980.

[4] R. M. Neal. Connectionist learning of belief networks. *Artificial Intelligence*, 56:71–113, 1992.

[5] P. Dayan, G. E. Hinton, R. M. Neal, and R. S. Zemel. Helmholtz machines. *Neural Computation*, 7:1022–1037, 1995.

[6] G. E. Hinton, P. Dayan, B. J. Frey, and R. M. Neal. The wake-sleep algorithm for self-organizing neural networks. *Science*, 268:1158–1161, 1995.

[7] L. K. Saul and M. I. Jordan. Attractor dynamics in feedforward neural networks. Submitted for publication.

[8] B. J. Frey, G. E. Hinton, and P. Dayan. Does the wake-sleep algorithm produce good density estimators? In D. Touretzky, M. Mozer, and M. Hasselmo, editors, *Advances in Neural Information Processing Systems*, volume 8. The MIT Press, 1995.

[9] M. R. Luettgen and A. S. Willsky. Likelihood calculation for a class of multiscale stochastic models, with application to texture discrimination. *IEEE Transactions on Image Processing*, 4(2):194–207, 1995.

[10] W. W. Irving, P. W. Fieguth, and A. S. Willsky. An overlapping tree approach to multiscale stochastic modeling and estimation. *IEEE Transactions on Image Processing*, 1995.

[11] R. S. Zemel, M. C. Mozer, and G. E. Hinton. TRAFFIC: Recognizing objects using hierarchical reference frame transformations. In *Advances in Neural Information Processing Systems*, volume 2. Morgan Kaufmann Publishers, San Mateo CA, 1990.

[12] P. Simard, Y. Le Cun, and J. Denker. Efficient pattern recognition using a new transformation distance. In S. Hanson, J. Cowan, and L. Giles, editors, *Advances in Neural Information Processing Systems*, volume 5. Morgan Kaufmann Publishers, San Mateo CA, 1992.

[13] G. E. Hinton, P. Dayan, and M. Revow. Modeling the manifolds of images of handwritten digits. *IEEE Transactions on Neural Networks*, 8:65–74, 1997.

[14] M. E. Tipping and C. M. Bishop. Mixtures of probabilistic principal component analysis. Technical Report NCRG/97/003, Aston University, Department of Computer Science and Applied Mathematics, 1997.

[15] A.P. Dempster, N.M. Laird, and D.B. Rubin. Maximum likelihood from incomplete data via the EM algorithm. *Journal of the Royal Statistical Society B*, 39:1–38, 1977.

[16] G. E. Hinton and M. Revow. Using mixtures of factor analyzers for segmentation and pose estimation, 1997.

[17] J. J. Hull. A database for handwritten text recognition research. *IEEE Transactions on Pattern Analysis and Machine Intelligence*, 16(5):550–554, 1994.
